# Branch and Bound for
# Semi-Supervised Support Vector Machines

**Olivier Chapelle**[1]
Max Planck Institute
Tübingen, Germany
chapelle@tuebingen.mpg.de

**Vikas Sindhwani**
University of Chicago
Chicago, USA
vikass@cs.uchicago.edu

**S. Sathiya Keerthi**
Yahoo! Research
Santa Clara, USA
selvarak@yahoo-inc.com

## Abstract

Semi-supervised SVMs ($S^3$VM) attempt to learn low-density separators by maximizing the margin over labeled and unlabeled examples. The associated optimization problem is non-convex. To examine the full potential of $S^3$VMs modulo local minima problems in current implementations, we apply branch and bound techniques for obtaining exact, *globally optimal* solutions. Empirical evidence suggests that the globally optimal solution can return excellent generalization performance in situations where other implementations fail completely. While our current implementation is only applicable to small datasets, we discuss variants that can potentially lead to practically useful algorithms.

## 1  Introduction

A major line of research on extending SVMs to handle partially labeled datasets is based on the following idea: solve the standard SVM problem while treating the unknown labels as additional optimization variables. By maximizing the margin in the presence of unlabeled data, one learns a decision boundary that traverses through low data-density regions while respecting labels in the input space. In other words, this approach implements the *cluster assumption* for semi-supervised learning – that points in a data cluster have similar labels. This idea was first introduced in [14] under the name *Transductive SVM*, but since it learns an inductive rule defined over the entire input space, we refer to this approach as *Semi-supervised SVM* ($S^3$VM).

Since its first implementation in [9], a wide spectrum of techniques have been applied to solve the non-convex optimization problem associated with $S^3$VMs, e.g., local combinatorial search [9], gradient descent [6], continuation techniques [3], convex-concave procedures [7], and deterministic annealing [12]. While non-convexity is partly responsible for this diversity of methods, it is also a departure from one of the nicest features of SVMs. Several experimental studies have established that $S^3$VM implementations show varying degrees of empirical success. This is conjectured to be closely tied to their susceptibility to local minima problems.

The following questions motivate this paper: How well do current $S^3$VM implementations approximate the exact, globally optimal solution of the non-convex problem associated with $S^3$VMs ? Can one expect significant improvements in generalization performance by better approaching the global solution? We believe that these questions are of fundamental importance for $S^3$VM research and are largely unresolved. This is partly due to the lack of simple implementations that practitioners can use to benchmark new algorithms against the global solution, even on small-sized problems.

Our contribution in this paper is to outline a class of Branch and Bound algorithms that are guaranteed to provide the globally optimal solution for $S^3$VMs. Branch and bound techniques have previously been noted in the context of $S^3$VM in [16], but no details were presented there. We implement and evaluate a branch and bound strategy that can serve as an upper baseline for $S^3$VM algorithms. This strategy is not practical for typical semi-supervised settings where large amounts of unlabeled data is available. But we believe it opens up new avenues of research that can potentially lead to more efficient variants.

Empirical results on some semi-supervised tasks presented in section 7 show that the exact solution found by branch and bound has excellent generalization performance, while other $S^3$VM implementations perform poorly. These results also show that $S^3$VM can compete and even outperform graph-based techniques (e.g.,[17, 13]) on problems where the latter class of methods have typically excelled.

## 2  Semi-Supervised Support Vector Machines

We consider the problem of binary classification. The training set consists of $l$ labeled examples $\{(\mathbf{x}_i, y_i)\}_{i=1}^l$, $y_i = \pm 1$, and of $u$ the unlabeled examples $\{\mathbf{x}_i\}_{i=l+1}^n$, with $n = l + u$. In the linear case, the following objective function is minimized on both the hyperplane parameters $\mathbf{w}$ and $b$, and on the label vector $\mathbf{y}_u := [y_{l+1} \dots y_n]^\top$,

$$\min_{\mathbf{w},b,\mathbf{y}_u,\xi_i \geq 0} \quad \frac{1}{2}\mathbf{w}^2 + C\sum_{i=1}^l \xi_i^p + C^* \sum_{i=l+1}^n \xi_i^p \tag{1}$$

under constraints $y_i(\mathbf{w} \cdot \mathbf{x}_i + b) \geq 1 - \xi_i, \quad 1 \leq i \leq n$. Non linear decision boundaries can be constructed using the *kernel trick* [15]. While in general any convex loss function can be used, it is common to either penalize the training errors linearly ($p = 1$) or quadratically ($p = 2$). In the rest of the paper, we consider $p = 2$. The first two terms in (1) correspond to a standard SVM. The last one takes into account the unlabeled points and can be seen as an implementation of the *cluster assumption* [11] or *low density separation* assumption [6]; indeed, it drives the outputs of the unlabeled points away from 0 (see figure 1).

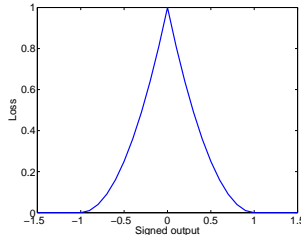

Figure 1: With $p = 2$ in (1), the loss of a point with label $y$ and signed output $t$ is $\max(0, 1 - yt)^2$. For an unlabeled point, this is $\min_y \max(0, 1 - yt)^2 = \max(0, 1 - |t|)^2$.

For simplicity, we take $C^* = C$. But in practice, it is important to set these two values independently because $C$ reflects our confidence in the labels of the training points, while $C^*$ corresponds to our belief in the low density separation assumption. In addition, we add the following balancing constraint to (1),

$$\frac{1}{u}\sum_{i=l+1}^n \max(y_i, 0) = r. \tag{2}$$

This constraint is necessary to avoid unbalanced solutions and has also been used in the original implementation [9]. Ideally, the parameter $r$ should be set to the ratio of positive points in the unlabeled set. Since it is unknown, $r$ is usually estimated through the class ratio on the labeled set. In that case, one may wish to "soften" this constraint, as in [6]. For the sake of simplicity, in the rest of the paper, we set $r$ to the true ratio of positive points in the unlabeled set.

Let us call $\mathcal{I}$ the objective function to be minimized:

$$\mathcal{I}(\mathbf{w}, b, \mathbf{y}_u) = \frac{1}{2}\mathbf{w}^2 + C\sum_{i=1}^{n}\max(0, 1 - y_i(\mathbf{w}\cdot\mathbf{x}_i + b))^2.$$

There are two main strategies to minimize $\mathcal{I}$:
(1) For a given fixed $\mathbf{w}$ and $b$, the optimal $\mathbf{y}_u$ is simply given by the signs of $\mathbf{w}\cdot\mathbf{x}_i + b$. Then a continuous optimization on $\mathbf{w}$ and $b$ can be done [6]. But note that the constraint (2) is then not straightforward to enforce.
(2) For a given $\mathbf{y}_u$, the optimization on $\mathbf{w}$ and $b$ is a standard SVM training. Let's define

$$\mathcal{J}(\mathbf{y}_u) = \min_{\mathbf{w}, b} \ \mathcal{I}(\mathbf{w}, b, \mathbf{y}_u). \tag{3}$$

Now the goal is to minimize $\mathcal{J}$ over a set of binary variables (and each evaluation of $\mathcal{J}$ is a standard SVM training). This was the approach followed in [9] and the one that we take in this paper. The constraint (2) is implemented by setting $\mathcal{J}(\mathbf{y}_u) = +\infty$ for all vectors $\mathbf{y}_u$ not satisfying it.

## 3 Branch and bound

### 3.1 Branch and bound basics

Suppose we want to minimize a function $f$ over a space $\mathcal{X}$, where $\mathcal{X}$ is usually discrete. A branch and bound algorithm has two main ingredients:

**Branching** : the region $\mathcal{X}$ is recursively split into smaller subregions. This yields a *tree* structure where each node corresponds to a subregion.

**Bounding** : consider two (disjoint) subregions (i.e. nodes) $A$ and $B \subset \mathcal{X}$. Suppose that an upper bound (say $a$) on the best value of $f$ over A is known and a lower bound (say $b$) on the best value of $f$ over $B$ is known and that $a < b$. Then, we know there is an element in the subset $A$ that is better than all elements of $B$. So, when searching for the global minimizer we can safely discard the elements of $B$ from the search: the subtree corresponding to $B$ is *pruned*.

### 3.2 Branch and bound for $S^3VM$

The aim is to minimize (3) over all $2^u$ possible choices for the vector $\mathbf{y}_u,$[1] which constitute the set $\mathcal{X}$ introduced above. The binary search tree has the following structure. Any node corresponds to a partial labeling of the data set and its two children correspond to the labeling of some unlabeled point. One can thus associate with any node a labeled set $L$ containing both the original labeled examples and a *subset $S$* of unlabeled examples $\{(\mathbf{x}_j, y_j)\}_{j\in S\subseteq[l+1...n]}$ to which the labels $y_j$ have been assigned. One can also associate an unlabeled set $U = [l + 1 \dots n] \setminus S$ corresponding to the subset of unlabeled points which have not been assigned a label yet. The size of the subtree rooted at this node is thus $2^{|U|}$. The root of the tree has only the original set of labeled examples associated with it, i.e $S$ is empty. The leaves in the tree correspond to a complete labeling of the dataset, i.e. $U$ is empty. All other nodes correspond to partial labelings.

As for any branch and bound algorithm, we have to decide about the following choices,

**Branching:** For a given node in the tree (i.e. a partial labeling of the unlabeled set), what should be its two children (i.e. which unlabeled point should be labeled next)?

**Bounding:** Which upper and lower bounds should be used?

**Exploration:** In which order will the search tree be examined? In other words, which subtree should be explored next? Note that the tree is not built explicitly but on the fly as we explore it.

Concerning the upper bound, we decided to have the following simple strategy: for a leaf node, the upper bound is simply the value of the function; for a non leaf node, there is no upper bound. In other words, the upper bound is the *best objective function found so far*. Coming back to the notations of section 3.1, the set $A$ is the leaf corresponding to the best solution found so far and the set $B$ is the subtree that we are considering to explore.

Because of this choice for the upper bound, a natural way to explore the tree is a *depth first search*. Indeed it is important to go to the leaves as often as possible in order to have a tight upper bound and thus perform aggressive pruning.

The choice of the lower bound and the branching strategy are presented next.

## 4   Lower bound

We consider a simple lower bound based on the following observation. The minimum of the objective function (1) is smaller when $C^* = 0$ than when $C^* > 0$. But $C^* = 0$ corresponds to a standard SVM, ignoring the unlabeled data. We can therefore compute a lower bound at a given node by optimizing a standard SVM on the labeled set associated with this node.

We now present a more general framework for computing lower bounds. It is based on the dual objective function of SVMs. Let $D(\boldsymbol{\alpha}, \mathbf{y}_U)$ be the dual objective function, where $\mathbf{y}_U$ corresponds to the labels of the unlabeled points which have not been assigned a label yet,

$$D(\boldsymbol{\alpha}, \mathbf{y}_U) = \sum_{i=1}^{n} \alpha_i - \frac{1}{2} \sum_{i,j=1}^{n} \alpha_i \alpha_j y_i y_j \left( K(\mathbf{x}_i, \mathbf{x}_j) + \frac{\delta_{ij}}{2C} \right). \tag{4}$$

The dual feasibility is

$$\alpha_i \geq 0 \quad \text{and} \quad \sum \alpha_i y_i = 0. \tag{5}$$

Now suppose that we have a strategy that, given $\mathbf{y}_U$, finds a vector $\boldsymbol{\alpha}(\mathbf{y}_U)$ satisfying (5). Since the dual is maximized,

$$D(\boldsymbol{\alpha}(\mathbf{y}_U), \mathbf{y}_U) \leq \max_{\boldsymbol{\alpha}} D(\boldsymbol{\alpha}, \mathbf{y}_U) = \mathcal{J}(\mathbf{y}_U),$$

where $\mathcal{J}$ has been defined in (3).

Let $Q(\mathbf{y}_U) := D(\boldsymbol{\alpha}(\mathbf{y}_U), \mathbf{y}_U)$ and *lb* a lower bound on (or the value of) $\min Q(\mathbf{y}_U)$, where the minimum is taken over all $\mathbf{y}_U$ satisfying the balancing constraint (2). Then *lb* is also a lower bound for the value of the objective function corresponding to that node.

The goal is thus to find a choice for $\boldsymbol{\alpha}(\mathbf{y}_U)$ such that a lower bound on $Q$ can be computed efficiently. The choice corresponding to the lower bound presented above is the following. Train an SVM on the labeled points, obtain the vector $\boldsymbol{\alpha}$ and complete it with zeros for the unlabeled points. Then $Q(\mathbf{y}_U)$ is the same for all the possible labelings of the unlabeled points and the lower bound is the SVM objective function on the labeled points.

Here is a sketch of another possibility for $\boldsymbol{\alpha}(\mathbf{y}_U)$ that one can explore: instead of completing the vector $\boldsymbol{\alpha}$ by zeros, we complete it by a constant $\gamma$ which would typically be of the same order of magnitude as $\boldsymbol{\alpha}$. Then $Q(\mathbf{y}_U) = \sum \alpha_i - \frac{1}{2} \mathbf{y}^\top H \mathbf{y}$, where $H_{ij} = \alpha_i \alpha_j K_{ij}$. To lower bound $Q$, one can use results from the quadratic zero-one programming literature [10] or solve a constrained eigenvalue problem [8]. Finally, note that unless $\sum_U y_i = 0$, the constraint $\sum \alpha_i y_i = 0$ will not be satisfied. One remedy is to train the supervised SVM with the constraint $\sum \alpha_i y_i = -\gamma \sum_U y_i = \gamma(n - 2ru + \sum_L y_i)$ (because of (2)). In the primal, this amounts to penalizing the bias term $b$.

## 5   Branching

At a given node, some unlabeled points have already been assigned a label. Which unlabeled point should be labeled next? Since our strategy is to reach a good solution as soon as possible (see last paragraph of section 3.2), it seems natural to assign the label that we are

the most confident about. A simple possibility would be to branch on the unlabeled point which is the nearest from another labeled point using a reliable distance metric.

But we now present a more principled approach based on the analysis of the objective value. We say that we are "confident" about a particular label of an unlabeled point when assigning the opposite label results in a big increase of the objective value: this partial solution would then be unlikely to lead to the optimal one.

Let us formalize this strategy. Remember from section 3.2 that a node is associated with a set $L$ of currently labeled examples and a set $U$ of unlabeled examples. Let $s(L)$ be the SVM objective function trained on the labeled set,

$$s(L) = \min_{\mathbf{w},b} \frac{1}{2}\mathbf{w}^2 + C \sum_{(\mathbf{x}_i,y_i)\in L} \max(0, 1 - y_i(\mathbf{w}\cdot\mathbf{x}_i + b))^2. \tag{6}$$

As discussed in the previous section, the lower bound is $s(L)$. Now our branching strategy consists in selecting the following point in $U$,

$$\arg\max_{\mathbf{x}\in U,\ y\in\pm 1} s(L\cup\{\mathbf{x},y\}) \tag{7}$$

In other words, we want to find the unlabeled point $\mathbf{x}^*$ and its label $y^*$ which would make the objective function increase as much as possible. Then we branch on $\mathbf{x}^*$, but start exploring the branch with the most likely label $-y^*$. This strategy has an intuitive link with the "label propagation" idea [17]: an unlabeled point which is near from a labeled point is likely to be of the same label; otherwise, the objective function would be large.

A main disadvantage of this approach is that to solve (7), a lot of SVM trainings are necessary. It is however possible to approximately compute $s(L\cup\{\mathbf{x},y\})$. The idea is similar to the fast approximation of the leave-one-out solution [5]. Here the situation is "add-one-in". If an SVM has been trained on the set $L$ it is possible to efficiently compute the solution when one point is added in the training set. This is under the assumption that the set of support vectors does not change when adding this point. In practice, the set is likely to change and the solution will only be approximate.

**Proposition 1** *Consider training an SVM on a labeled set $L$ with quadratic penalization of the errors (cf (6) or (4)). Let f be the learned function and* sv *be the set of support vectors. Then, if* sv *does not change while adding a point $(\mathbf{x}, y)$ in the training set,*

$$s(L\cup\{\mathbf{x},y\}) = s(L) + \frac{\max(0, 1 - yf(\mathbf{x}))^2}{2S_{\mathbf{x}}^2 + 1/C} \tag{8}$$

*where $S_{\mathbf{x}}^2 = K(\mathbf{x},\mathbf{x}) - \mathbf{v}^\top K_{sv}^{-1}\mathbf{v}$,*
$$K_{sv} = \begin{pmatrix} \left(K(\mathbf{x}_i,\mathbf{x}_j) + \frac{\delta_{ij}}{2C}\right)_{i,j\in sv} & \mathbf{1} \\ \mathbf{1}^\top & 0 \end{pmatrix} \text{ and } \mathbf{v}^\top = (\tilde{K}(\mathbf{x}_i,\mathbf{x})_{i\in sv}\ \ 1).$$

The proof is omitted because of lack of space. It is based on the fact that $s(L) = \frac{1}{2}\mathbf{y}_{sv}^\top K_{sv}^{-1}\mathbf{y}_{sv}$ and relies on the block matrix inverse formula.

# 6  Algorithm

The algorithm is implemented recursively (see algorithm 1). At the beginning, the upper bound can either be set to $+\infty$ or to a solution found by another algorithm.

Note that the SVM trainings are incremental: whenever we go down the tree, one point is added in the labeled set. For this reason, the retraining can be done efficiently (also see [2]) since effectively, we just need to update the inverse of a matrix.

# 7  Experiments

We consider here two datasets where other $S^3$VM implementations are unable to achieve satisfying test error rates. This naturally raises the following questions: Is this weak per-

---

**Algorithm 1** Branch and bound for S³VM(BB).

---

**Function:** $(Y^*, v) \leftarrow$ S³VM$(Y, ub)$ &emsp;&emsp;&emsp;&emsp;&emsp; % Recursive implementation
**Input:** &emsp; $Y$: a partly labeled vector (0 for unlabeled)
&emsp;&emsp;&emsp;&emsp; $ub$: an upper bound on the optimal objective value.
**Output:** &emsp; $Y^*$: optimal fully labeled vector
&emsp;&emsp;&emsp;&emsp; $v$: corresponding objective function.
&emsp;**if** $\sum \max(0, Y_i) > ur$ &emsp; OR &emsp; $\sum \max(0, -Y_i) < n - ur$ **then**
&emsp;&emsp;**return** &emsp;&emsp;&emsp;&emsp;&emsp;&emsp;&emsp;&emsp;&emsp; % Constraint (2) can not be satisfied
&emsp;**end if** &emsp;&emsp;&emsp;&emsp;&emsp;&emsp;&emsp;&emsp;&emsp;&emsp;&emsp; $\longrightarrow$ Do not explore this subtree
&emsp;$v \leftarrow$ SVM$(Y)$ &emsp;&emsp;&emsp; % Compute the SVM objective function on the labeled points.
&emsp;**if** $v > ub$ **then**
&emsp;&emsp;**return** &emsp;&emsp;&emsp;&emsp;&emsp;&emsp;&emsp; % The lower bound is higher than the upper bound
&emsp;**end if** &emsp;&emsp;&emsp;&emsp;&emsp;&emsp;&emsp;&emsp;&emsp;&emsp;&emsp; $\longrightarrow$ Do not explore this subtree
&emsp;**if** $Y$ is fully labeled **then**
&emsp;&emsp;$Y^* \leftarrow Y$
&emsp;&emsp;**return** &emsp;&emsp;&emsp;&emsp;&emsp;&emsp;&emsp;&emsp;&emsp;&emsp;&emsp;&emsp; % We are at a leaf
&emsp;**end if**
&emsp;Find index $i$ and label $y$ as in (7) &emsp;&emsp;&emsp; % Find next unlabeled point to label
&emsp;$Y_i \leftarrow -y$ &emsp;&emsp;&emsp;&emsp;&emsp;&emsp;&emsp;&emsp;&emsp; % Start first by the most likely label
&emsp;$(Y^*, v) \leftarrow$ S³VM$(Y, ub)$ &emsp;&emsp;&emsp;&emsp; % Find (recursively) the best solution
&emsp;$Y_i \leftarrow -Y_i$ &emsp;&emsp;&emsp;&emsp;&emsp;&emsp;&emsp;&emsp;&emsp;&emsp; % Switch the label
&emsp;$(Y_2^*, v_2) \leftarrow$ S³VM$(Y, \min(ub, v))$ &emsp; % Explore other branch with updated upper-bound
&emsp;**if** $v_2 < v$ **then**
&emsp;&emsp;$Y^* \leftarrow Y_2^*$ and $v \leftarrow v_2$ &emsp;&emsp;&emsp;&emsp;&emsp;&emsp;&emsp; % Keep the best solution
&emsp;**end if**

---

formance due to the unsuitability of the S³VM objective function for these problems or do these methods get stuck at highly sub-optimal local minima?

## 7.1 Two moons

The "two moons" dataset is now a standard benchmark for semi-supervised learning algorithms. Most graph-based methods such as [17] easily solve this problem , but so far, all S³VM algorithms find it difficult to construct the right boundary (an exception is [12] using an $L_1$ loss). We drew 100 random realizations of this dataset, fixed the bandwidth of an RBF kernel to $\sigma = 0.5$ and set $C = 10$. Each moon contained 50 unlabeled points.

We compared $\nabla$S³VM[6], cS³VM[3], CCCP [7], SVM$^{light}$ [9] and DA [12]. For the first 3 methods, there is no direct way to enforce the constraint (2). However, these methods have a constraint that the mean output on the unlabeled point should be equal to some constant. This constant is normally fixed to the mean of the labels, but for the sake of consistency we did a dichotomy search on this constant in order to have (2) satisfied.

Results are presented in table 1. Note that the test errors for other S³VM implementations are likely to be improved by hyperparameter tuning, but they will still stay very high. For comparison, we have also included the results of a state-of-the-art graph based method, LapSVM [13] whose hyperparameters were optimized for the test error and the threshold adjusted to satisfy the constraint (2).

Matlab source code and a demo of our algorithm on the "two moons" dataset is accessible as supplementary material with this paper.

## 7.2 COIL

Extensive benchmark results reported in [4, benchmark chapter] show that on problems where classes are expected to reside on low-dimensional non-linear manifolds, e.g., handwritten digits, graph-based algorithms significantly outperform S³VM implementations.

Table 1: Results on the two moons dataset (averaged over 100 random realizations)

|  | Test error (%) | Objective function |
|---|---|---|
| $\nabla$S$^3$VM | 59.3 | 13.64 |
| cS$^3$VM | 45.7 | 13.25 |
| CCCP | 64 | 39.55 |
| SVM$^{light}$ | 66.2 | 20.94 |
| DA | 34.1 | 46.85 |
| BB | 0 | 7.81 |
| LapSVM | 3.7 | N/A |

We consider here such a dataset by selecting three confusible classes from the COIL20 dataset [6] (see figure 2). There are 72 images per class, corresponding to rotations of 5 degrees (and thus yielding a one dimensional manifold). We randomly selected 2 images per class to be in the labeled set and the rest being unlabeled. Results are reported in table 2. The hyperparameters were chosen to be $\sigma = 3000$ and $C = 100$.

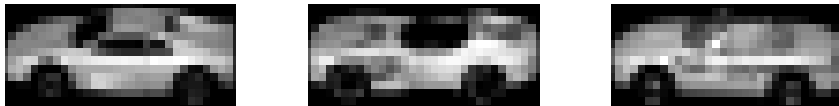

Figure 2: The 3 cars from the COIL dataset, subsampled to 32×32

Table 2: Results on the Coil dataset (averaged over 10 random realizations)

|  | Test error (%) | Objective function |
|---|---|---|
| $\nabla$S$^3$VM | 60.6 | 267.4 |
| cS$^3$VM | 60.6 | 235 |
| CCCP | 47.5 | 588.3 |
| SVM$^{light}$ | 55.3 | 341.6 |
| DA | 48.2 | 611 |
| BB | 0 | 110.7 |
| LapSVM | 7.5 | N/A |

From tables 1 and 2, it appears clearly that (1) the S$^3$VM objective function leads to excellent test errors; (2) other S$^3$VM implementations fail completely in finding a good minimum of the objective function[2] and (3) the global S$^3$VM solution can actually outperform graph-based alternatives even if other S$^3$VM implementations are not found to be competitive.

Concerning the running time, it is of the order of a minute for both datasets. We do not expect this algorithm to be able to handle datasets much larger than couple of hundred points.

## 8   Discussion and Conclusion

We implemented and evaluated one strategy amongst many in the class of branch and bound methods to find the *globally optimal* solution of S$^3$VMs. The work of [1] is the most closely related to our methods. However that paper presents an algorithm for *linear* S$^3$VMs and relies on generic mixed integer programming which does not make use of the problem structure as our methods can.

This basic implementation can perhaps be made more efficient by choosing better bounding and branching schemes. Also, by fixing the upper bound as the currently best objective

value, we restricted our implementation to follow depth-first search. It is conceivable that breadth-first search is equally or more effective in conjunction with alternative upper bounding schemes. Pruning can be done more aggressively to speed-up termination at the expense of obtaining a solution that is suboptimal within some tolerance (i.e prune B if $a < b - \epsilon$). Finally, we note that a large family of well-tested branch and bound procedures from zero-one quadratic programming literature can be immediately applied to the S³VM problem for the special case of squared loss. An interesting open question is whether one can provide a guarantee for polynomial time convergence under some assumptions on the data and the kernel.

Concerning the running time of our current implementation, we have observed that it is most efficient whenever the global minimum is significantly smaller than most local minima: in that case, the tree can be pruned efficiently. This happens when the clusters are well separated and $C$ and $\sigma$ are not too small.

For these reasons, we believe that this implementation does not scale to large datasets, but should instead be considered as a proof of concept: the S³VM objective function is very well suited for semi-supervised learning and more effort should be made on trying to efficiently find good local minima.

## Footnotes

[1]Now part of Yahoo! Research, chap@yahoo-inc.com

[1]There are actually only $\begin{pmatrix} u \\ ur \end{pmatrix}$ effective choices because of the constraint (2).

[2]The reported test errors are somehow irrelevant and should not be used for ranking the different algorithms. They should just be interpreted as "failure".

# References

[1] K. Bennett and A. Demiriz. Semi-supervised support vector machines. In *Advances in Neural Information processing systems 12*, 1998.

[2] G. Cauwenberghs and T. Poggio. Incremental and decremental support vector machine learning. In *Advances in Neural Information Processing Systems*, pages 409–415, 2000.

[3] O. Chapelle, M. Chi, and A. Zien. A continuation method for semi-supervised svms. In *International Conference on Machine Learning*, 2006.

[4] O. Chapelle, B. Schölkopf, and A. Zien, editors. *Semi-Supervised Learning*. MIT Press, Cambridge, 2006. in press. `www.kyb.tuebingen.mpg.de/ssl-book/`.

[5] O. Chapelle, V. Vapnik, O. Bousquet, and S. Mukherjee. Choosing multiple parameters for support vector machines. *Machine Learning*, 46:131–159, 2002.

[6] O. Chapelle and A. Zien. Semi-supervised classification by low density separation. In *Tenth International Workshop on Artificial Intelligence and Statistics*, 2005.

[7] R. Collobert, F. Sinz, J. Weston, and L. Bottou. Large scale transductive SVMs. *Journal of Machine Learning Research*, 7:1687–1712, 2006.

[8] W. Gander, G. H. Golub, and U. Von Matt. A constrained eigenvalue problem. *Linear Algebra and its Applications*, 114/115:815–839, 1989.

[9] T. Joachims. Transductive inference for text classification using support vector machines. In *International Conference on Machine Learning*, 1999.

[10] P.M. Pardalos and G.P. Rodgers. Computational aspects of a branch and bound algorithm for quadratic zero-one programming. *Computing*, 45:131–144, 1990.

[11] M. Seeger. A taxonomy of semi-supervised learning methods. In O. Chapelle, B. Schölkopf, and A. Zien, editors, *Semi-Supervised Lerning*. MIT Press, 2006.

[12] V. Sindhwani, S. Keerthi, and O. Chapelle. Deterministic annealing for semi-supervised kernel machines. In *International Conference on Machine Learning*, 2006.

[13] V. Sindhwani, P. Niyogi, and M. Belkin. Beyond the point cloud: From transductive to semi-supervised learning. In *International Conference on Machine Learning*, 2005.

[14] V. Vapnik and A. Sterin. On structural risk minimization or overall risk in a problem of pattern recognition. *Automation and Remote Control*, 10(3):1495–1503, 1977.

[15] V. N. Vapnik. *Statistical Learning Theory*. John Wiley & Sons, Inc., New York, 1998.

[16] W. Wapnik and A. Tscherwonenkis. *Theorie der Zeichenerkennung*. Akademie Verlag, Berlin, 1979.

[17] X. Zhu and Z. Ghahramani. Learning from labeled and unlabeled data with label propagation. Technical Report 02-107, CMU-CALD, 2002.
